# An Integrated Vision Sensor for the Computation of Optical Flow Singular Points

**Charles M. Higgins and Christof Koch**

Division of Biology, 139-74
California Institute of Technology
Pasadena, CA 91125
`[chuck,koch]@klab.caltech.edu`

## Abstract

A robust, integrative algorithm is presented for computing the position of the focus of expansion or axis of rotation (the singular point) in optical flow fields such as those generated by self-motion. Measurements are shown of a fully parallel CMOS analog VLSI motion sensor array which computes the direction of local motion (sign of optical flow) at each pixel and can directly implement this algorithm. The flow field singular point is computed in real time with a power consumption of less than $2\ mW$. Computation of the singular point for more general flow fields requires measures of field expansion and rotation, which it is shown can also be computed in real-time hardware, again using only the sign of the optical flow field. These measures, along with the location of the singular point, provide robust real-time self-motion information for the visual guidance of a moving platform such as a robot.

## 1   INTRODUCTION

Visually guided navigation of autonomous vehicles requires robust measures of self-motion in the environment. The heading direction, which corresponds to the focus of expansion in the visual scene for a fixed viewing angle, is one of the primary sources of guidance information. Psychophysical experiments [WH88] show that humans can determine their heading direction very precisely. In general, the location of the singular point in the visual field provides important self-motion information.

Optical flow, representing the motion seen in each local area of the visual field, is partic-

ularly compute-intensive to process in real time. We have previously shown [DHK97] a fully parallel, low power, CMOS analog VLSI vision processor for computing the local direction of motion. With onboard photoreceptors, each pixel computes in continuous time a vector corresponding to the sign of the local normal flow. In this article, we show how these motion vectors can be integrated in hardware to compute the singular point of the optical flow field. While each individual pixel suffers from transistor mismatch and spatial variability with respect to its neighbors, the integration of many pixels serves to average out these irregularities and results in a highly robust computation. This compact, low power self-motion processor is well suited for autonomous vehicle applications.

Extraction of self-motion information has been a topic of research in the machine vision community for decades, and has generated volumes of research; see [FA97] for a good review. While many algorithms exist for determining flow field singular points in complex self-motion situations, few are suitable for real-time implementation. Integrated hardware attempts at self-motion processing have only begun recently, with the work of Indiveri et al [IKK96]. The zero crossing in a 1D array of CMOS velocity sensors was used to detect one component of the focus of expansion. In a separate chip, the sum of a radial array of velocity sensors was used to compute the rate of flow field expansion, from which the time-to-contact can be calculated. McQuirk [McQ96] built a CCD-based image processor which used an iterative algorithm to locate consistent stable points in the image, and thus the focus of expansion. More recently, Deutschmann et al. [DW98] have extended Indiveri et al.'s work to 2D by summing rows and columns in a 2D CMOS motion sensor array and using software to detect zero crossings and find the flow field singular point.

## 2  SINGULAR POINT ALGORITHM

In order to compute the flow field singular point, we compute the sum of the *sign* of optical flow over the entire field of view. Let the field of view be centered at $(0,0)$ and bounded by $\pm L$ in both spatial dimensions; then (vector quantities are indicated in boldface)

$$\mathbf{S} = \int_{-L}^{L} \int_{-L}^{L} \mathbf{U}(x,y)\, dx\, dy \tag{1}$$

where $\mathbf{U}(x,y) = (U_x(x,y), U_y(x,y)) = \operatorname{sgn}(\mathbf{V}(x,y))$ and $\mathbf{V}(x,y)$ is the optical flow field. Consider a purely expanding flow field with the focus of expansion (FOE) at the center of the visual field. Intuitively, the vector sum of the sign of optical flow will be zero, because each component is balanced by a spatially symmetric component with opposite sign. As the FOE moves away from the center of the visual field, the sum will increase or decrease depending on the FOE position.

An expanding flow field may be expressed as

$$\mathbf{V_e}(x,y) = A(x,y) \cdot ((x - X_e),(y - Y_e)) \tag{2}$$

where $A(x,y)$ denotes the local rate of expansion and $(X_e, Y_e)$ is the focus of expansion. The integral (1) applied to this flow field yields

$$\mathbf{S} = -4L \cdot (X_e, Y_e)$$

as long as $A$ is positive. Note that, due to the use of optical flow *sign* only, this quantity is independent of the speed of the flow field components. We will discuss in Section 5 how the positivity requirement of $A$ can be relaxed somewhat.

Similarly, a clockwise rotating flow field may be expressed as

$$\mathbf{V_r}(x,y) = B(x,y) \cdot ((y - Y_r), -(x - X_r)) \tag{3}$$

where $B(x,y)$ denotes the local rate of rotation and $(X_r, Y_r)$ is the axis of rotation (AOR). The integral (1) applied to this flow field yields

$$\mathbf{S} = -4L \cdot (Y_r, -X_r)$$

as long as $B$ is positive.

Let us now consider the case of a combination of these expanding and rotating fields (2) and (3):

$$\mathbf{V}(x,y) = \theta \mathbf{V_e} + (1 - \theta)\mathbf{V_r} \tag{4}$$

This flow field is spiral in shape; the parameter $\theta$ defines the mix of the two field types. The sum in this case is more complex to evaluate, but for $\theta$ small (rotation dominating),

$$\mathbf{S} = -4L \cdot (CX_e + Y_r, CY_e - X_r) \tag{5}$$

and for $\theta$ large (expansion dominating),

$$\mathbf{S} = -4L \cdot (X_e + (1/C)Y_r, Y_e - (1/C)X_r) \tag{6}$$

where $C = \frac{\theta A}{1 - \theta B}$. Since it is mathematically impossible to recover *both* the FOE and AOR with only two equations,[1] let us equate the FOE and AOR and concentrate on recovering the unique singular point of this spiral flow field. In order to do this, we need a measurement of the quantity $C$, which reflects the relative mix and strength of the expanding and rotating flow fields.

## 2.1 COEFFICIENTS OF EXPANSION AND ROTATION

Consider a contour integral around the periphery of the visual field of the sign of optical flow components *normal* to the contour of integration. If we let this contour be a square of size $2L$ centered at $(0,0)$, we can express this integral as

$$8LC_{exp} = \int_{-L}^{L} (U_y(x, L) - U_y(x, -L))\, dx + \int_{-L}^{L} (U_x(L, y) - U_x(-L, y))\, dy \tag{7}$$

This integral can be considered as a 'template' for expanding flow fields. The quantity $C_{exp}$ reaches unity for a purely expanding flow field with FOE within the visual field, and reaches zero for a purely rotating flow field. A similar quantity for rotation may be defined by an integral of the sign of optical flow components *parallel* to the contour of integration:

$$8LC_{rot} = \int_{-L}^{L} (U_x(x, L) - U_x(x, -L))\, dx + \int_{-L}^{L} (U_y(-L, y) - U_y(L, y))\, dy \tag{8}$$

It can be shown that for $\theta$ small (rotation dominating), $C_{exp} \approx C$. As $\theta$ increases, $C_{exp}$ saturates at unity. Similarly, for $\theta$ large (expansion dominating), $C_{rot} \approx (1/C)$. As $\theta$ decreases, $C_{rot}$ saturates at unity. This suggests the following approximation to equations (5) and (6), letting $X_s = X_e = X_r$ and $Y_s = Y_e = Y_r$

$$\mathbf{S} = -4L \cdot (C_{exp}X_s + C_{rot}Y_s, C_{exp}Y_s - C_{rot}X_s) \tag{9}$$

from which equation the singular point $(X_s, Y_s)$ may be uniquely calculated. Note that this generalized expression also covers contracting and counterclockwise rotating fields (for which the quantities $C_{exp}$ and $C_{rot}$ would be negative).

## 3  HARDWARE IMPLEMENTATION

The real-time hardware implementation of the above algorithm utilizes a fully parallel 14×13 CMOS analog VLSI motion sensor array. The elementary motion detectors are briefly described below. Each pixel in the array creates a local motion vector when crossed by a spatial edge; this vector is represented by two currents encoding the $x$ and $y$ components. These currents persist for an adjustable period of time after stimulation. By using the serial pixel scanners at the periphery of the chip (normally used to address each pixel individually), it is possible to connect all of these currents to the same output wire, thus implementing the sum required by the algorithm. In this mode, the current outputs of the chip directly represent the sum $S$ in equation (1), and power consumption is less than $2\ mW$.

A similar sum combining sensor row and column outputs around the periphery of the chip could be used to implement the quantities $C_{exp}$ and $C_{rot}$ in equations (7) and (8). Due to the sign changes necessary, this sum cannot be directly implemented with the present implementation. However, it is possible to emulate this sum by scanning off the vector field and performing the sum in real-time software.

### 3.1  ELEMENTARY MOTION DETECTOR

The 1D elementary motion detector used in this processor is the ITI (Inhibit, Trigger, and Inhibit) sensor. Its basic operation is described in Figure 1; see [DHK97] for details. The sensor is edge sensitive, approximately invariant to stimulus contrast above 20% and functions over a stimulus velocity range from 10-800 pixels/sec.

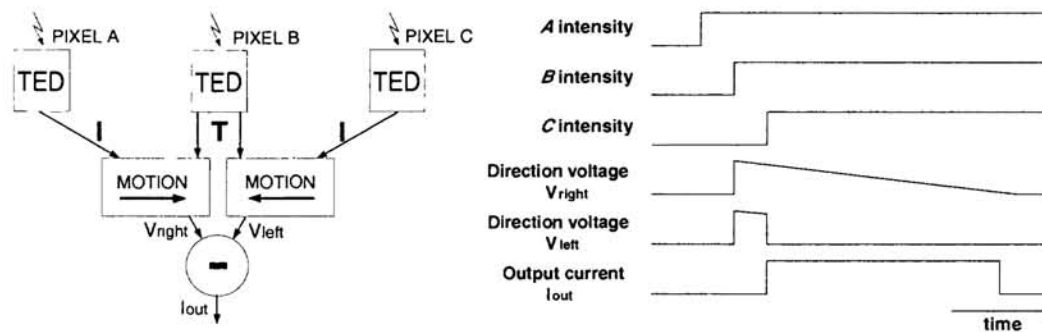

Figure 1: ITI sensor: a spatial edge crossing the sensor from left to right triggers direction voltages for both directions $V_{right}$ and $V_{left}$ in pixel $B$. The same edge subsequently crossing pixel $C$ inhibits the null direction voltage $V_{left}$. The output current is continuously computed as the difference between $V_{right}$ and $V_{left}$; the resulting positive output current $I_{out}$ indicates rightward motion. Pixels $B$ and $A$ interact similarly to detect leftward motion, resulting in a negative output current.

The output of each 1D ITI sensor represents the order in which the three involved photoreceptors were crossed by a spatial edge. Like all local motion sensors, it suffers from the aperture problem, and thus can only respond to the optical flow normal to the local gradients of intensity. The final result of this computation is the sign of the projection of the normal flow onto the sensor orientation. Two such sensors placed orthogonally effectively compute the sign of the normal flow vector.

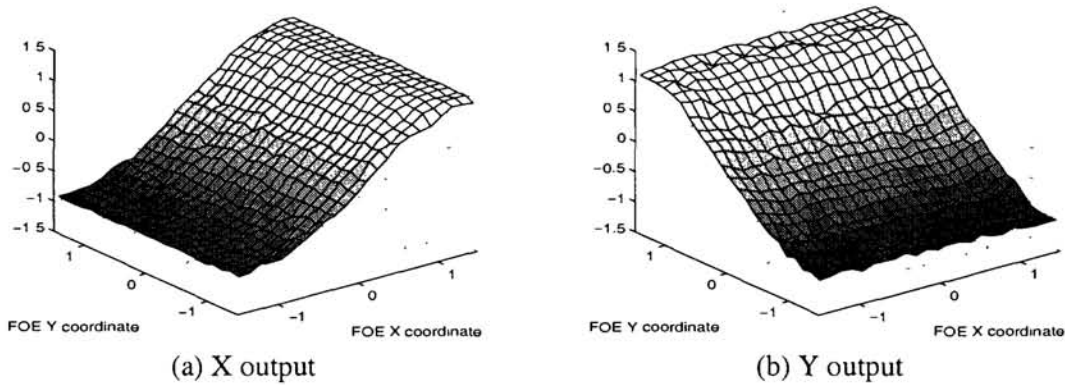

(a) X output                             (b) Y output

Figure 2: Hardware FOE computation: the chip was presented with a computer-generated image of high-contrast expanding circles; the FOE location was varied under computer control on a 2D grid. The measured chip current output has been scaled by a factor of $6 \times 10^5$ chip radii per Ampere. All FOE locations are shown in chip radii, where a radius of 1.0 corresponds to the periphery of the sensor array. Data shown is the mean output over one stimulus period; RMS variation is 0.27 chip radii.

## 4 SENSOR MEASUREMENTS

In Figure 2, we demonstrate the hardware computation of the FOE. To generate this data, the chip was presented with a computer-generated image of high-contrast expanding circles. The focus of expansion was varied on a 2D grid under computer control, and the mean of the chip's output current over one period of the stimulus was calculated for each FOE position. This output varies periodically with the stimulus because each motion sensor stops generating output while being crossed by a stimulus edge. The RMS value of this variation for the expanding circles stimulus is 0.27 chip radii; this variation can be decreased by increasing the resolution of the sensor array. The data shows that the FOE is precisely located when it is within the chip's visual field. Each component of the chip output is virtually independent of the other. When the FOE is outside the chip's visual field, the chip output saturates, but continues to indicate the correct direction towards the FOE.

The chip's AOR response to a rotating 'wagon wheel' stimulus is qualitatively and quantitatively very similar, and is not shown for lack of space.

In Figure 3, the coefficients of expansion and rotation are shown for the same expanding circles stimulus used in Figure 2. Since these coefficients cannot be calculated directly by the present hardware, the flow field was scanned out of the chip and these quantities were calculated in real-time software. While the FOE is on the chip, $C_{exp}$ remains near unity, dropping off as the FOE leaves the chip. As expected, $C_{rot}$ remains near zero regardless of the FOE position. Note that, because these coefficients are calculated by integrating a ring of only 48 sensors near the chip periphery, they have more spatial noise than the FOE calculation which integrates all 182 motion sensors.

In Figure 4, a spiral stimulus is presented, creating an equal combination of expansion and rotation ($\theta = 0.5$ in equation (4)). The singular point is calculated from equation (9) using the optical flow field scanned from the chip. Due to the combination of the coefficients with the sum computation, more spatial noise has been introduced than was seen in the FOE case. However, the singular point is still clearly located when within the chip. When the

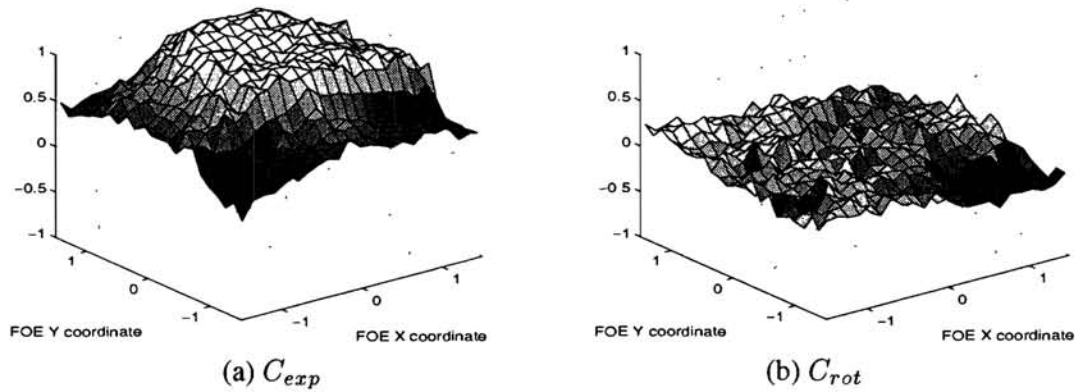

(a) $C_{exp}$                                                    (b) $C_{rot}$

Figure 3: Coefficients of expansion and rotation: again using the computer-generated expanding circles stimulus, the FOE was varied on a 2D grid. All FOE locations are shown in chip radii, where a radius of 1.0 corresponds to the periphery of the sensor array. Data shown is the mean output over one stimulus period.

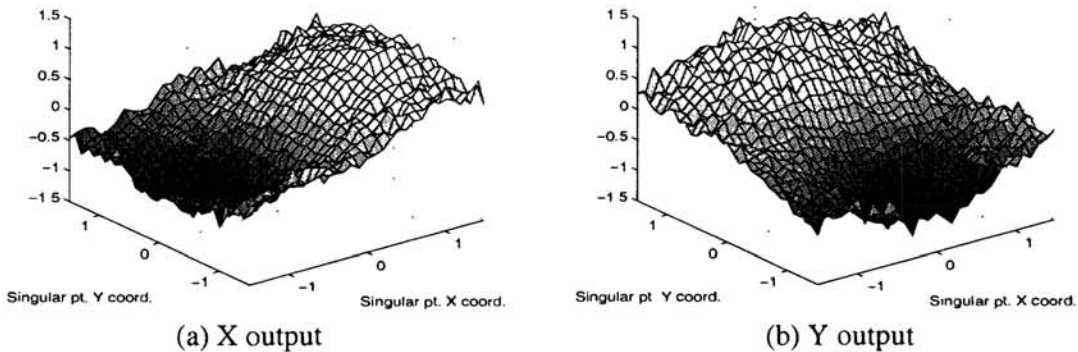

(a) X output                                                    (b) Y output

Figure 4: Singular point calculation: the chip was shown a computer-generated image of a rotating spiral; the singular point location was varied under computer control on a 2D grid. All singular point locations are shown in chip radii, where a radius of 1.0 corresponds to the periphery of the sensor array. Data shown is the mean output over one stimulus period.

singular point leaves the chip, the calculated position drops towards zero as the algorithm can no longer compute the mix of expansion and rotation.

## 5   DISCUSSION

We have presented a simple, robust algorithm for computing the singular point of an optical flow field and demonstrated a real-time hardware implementation. Due to the use of the *sign* of optical flow only, the solution is independent of the relative velocities of components of the flow field. Because a large number of individual sensors are integrated to produce this output, it is quite robust to the spatial variability of the individual motion sensors. We have also shown how coefficients indicating the mix of expansion and rotation may be computed in hardware. A motion sensor array which directly computes these coefficients, as well as the flow field singular point, is currently in fabrication.

In order to derive the equations relating the flow field sums to the FOE, it was necessary in Section 2 to make the unrealistic assumption that the optical flow field contains no areas of zero optical flow. Due to the persistence time of the motion sensor used, it is possible to relax this assumption significantly. As long as all parts of the visual field receive stimulation *within the persistence time of the motion output*, the optical flow field seen by the motion sensor array will contain no zeros and the singular point output will remain correct. This is a simple example of temporal motion integration. In fact, it is possible in practice to relax this assumption even further: as long as the location of zeros in the optical flow field is spatially random, the magnitude of the output will be reduced but it will continue to provide a clear error signal pointing towards the flow field singular point.

Because of the fully parallel design of the motion sensor array, larger arrays may be obtained by simply replicating pixels. The FOE summing algorithm is not affected by this increase in the number of pixels. As the number of pixels is increased, the average power consumption will increase sublinearly, because the sum output current (the dominant source of prolonged power consumption) can be maintained at approximately the same absolute value regardless of the number of pixels integrated. However, the periodic variation of the output with the stimulus will be decreased, the precision of the FOE output will be improved, and the need for temporal averaging will be reduced.

## Acknowledgments

This research was supported by the Caltech Center for Neuromorphic Systems Engineering as a part of the National Science Foundation's Engineering Research Center program, as well as by the Office of Naval Research. The authors wish to thank Rainer Deutschmann for stimulating discussions.

## Footnotes

[1] In fact, if $A$ and $B$ are constant, there exists no unique solution for the FOE and AOR.

## References

[DHK97]  R. Deutschmann, C. Higgins, and C. Koch. Real-time analog VLSI sensors for 2-D direction of motion. In *Proceedings of the Int. Conf. on Artificial Neural Networks*, pages 1163–1168. Springer Verlag, 1997.

[DW98]  R. A. Deutschmann and O. G. Wenisch. Compressive computation in analog VLSI motion sensors. In *Proceedings of Deutsche Arbeitsgemeinschaft für Mustererkennung*, 1998.

[FA97]  C. Fermüller and Y. Aloimonos. On the geometry of visual correspondence. *International Journal of Computer Vision*, 21(3):233–247, 1997.

[IKK96]  G. Indiveri, J. Kramer, and C. Koch. Parallel analog VLSI architectures for computation of heading direction and time-to-contact. In D.S. Touretzky, M.C. Mozer, and M.E. Hasselmo, editors, *Advances in Neural Information Processing Systems*, volume 8, pages 720–726, Cambridge, MA, 1996. MIT.

[McQ96]  I. McQuirk. An analog VLSI chip for estimating the focus of expansion. Technical Report 1577, Massachusetts Institute of Technology, Artificial Intelligence Laboratory, 1996.

[WH88]  W. Warren and D. Hannon. Direction of self-motion is perceived from optical-flow. *Nature*, 336(6195):162–163, 1988.